# Lower bounds on minimax rates for nonparametric regression with additive sparsity and smoothness

**Garvesh Raskutti**[1]**, Martin J. Wainwright**[1,2]**, Bin Yu**[1,2]
[1]UC Berkeley Department of Statistics
[2]UC Berkeley Department of Electrical Engineering and Computer Science

## Abstract

We study minimax rates for estimating high-dimensional nonparametric regression models with sparse additive structure and smoothness constraints. More precisely, our goal is to estimate a function $f^* : \mathbb{R}^p \to \mathbb{R}$ that has an additive decomposition of the form $f^*(X_1, \ldots, X_p) = \sum_{j \in S} h_j^*(X_j)$, where each component function $h_j^*$ lies in some class $\mathcal{H}$ of "smooth" functions, and $S \subset \{1, \ldots, p\}$ is an unknown subset with cardinality $s = |S|$. Given $n$ i.i.d. observations of $f^*(X)$ corrupted with additive white Gaussian noise where the covariate vectors $(X_1, X_2, X_3, ..., X_p)$ are drawn with i.i.d. components from some distribution $\mathbb{P}$, we determine lower bounds on the minimax rate for estimating the regression function with respect to squared-$L^2(\mathbb{P})$ error. Our main result is a lower bound on the minimax rate that scales as $\max\left(\frac{s \log(p/s)}{n}, s\,\epsilon_n^2(\mathcal{H})\right)$. The first term reflects the sample size required for performing *subset selection*, and is independent of the function class $\mathcal{H}$. The second term $s\,\epsilon_n^2(\mathcal{H})$ is an *s-dimensional estimation* term corresponding to the sample size required for estimating a sum of $s$ univariate functions, each chosen from the function class $\mathcal{H}$. It depends linearly on the sparsity index $s$ but is independent of the global dimension $p$. As a special case, if $\mathcal{H}$ corresponds to functions that are $m$-times differentiable (an $m^{th}$-order Sobolev space), then the $s$-dimensional estimation term takes the form $s\epsilon_n^2(\mathcal{H}) \asymp s\,n^{-2m/(2m+1)}$. Either of the two terms may be dominant in different regimes, depending on the relation between the sparsity and smoothness of the additive decomposition.

## 1  Introduction

Many problems in modern science and engineering involve high-dimensional data, by which we mean that the ambient dimension $p$ in which the data lies is of the same order or larger than the sample size $n$. A simple example is parametric linear regression under high-dimensional scaling, in which the goal is to estimate a regression vector $\beta^* \in \mathbb{R}^p$ based on $n$ samples. In the absence of additional structure, it is impossible to obtain consistent estimators unless the ratio $p/n$ converges to zero which precludes the regime $p \gg n$. In many applications, it is natural to impose sparsity conditions, such as requiring that $\beta^*$ have at most $s$ non-zero parameters for some $s \ll p$. The method of $\ell_1$-regularized least squares, also known as the Lasso algorithm [14], has been shown to have a number of attractive theoretical properties for such high-dimensional sparse models (e.g., [1, 19, 10]).

Of course, the assumption of a parametric linear model may be too restrictive for some applications. Accordingly, a natural extension is the non-parametric regression model $y = f^*(x_1, \ldots, x_p) + w$, where $w \sim N(0, \sigma^2)$ is additive observation noise. Unfortunately, this general non-parametric model is known to suffer severely from the "curse of dimensionality", in that for most natural function classes, the sample size $n$ required to achieve a given estimation accuracy grows exponentially in the dimension. This challenge motivates the use of additive non-parametric models (see the book [6] and references therein), in which the function $f^*$ is decomposed additively as a sum $f^*(x_1, x_2, ..., x_p) = \sum_{j=1}^p h_j^*(x_j)$ of univariate functions $h_j^*$. A natural sub-class of these

models are the *sparse additive models*, studied by Ravikumar et. al [12], in which

$$f^*(x_1, x_2, ..., x_p) = \sum_{j \in S} h_j^*(x_j), \tag{1}$$

where $S \subset \{1, 2, \ldots, p\}$ is some *unknown* subset of cardinality $|S| = s$.

A line of past work has proposed and analyzed computationally efficient algorithms for estimating regression functions of this form. Just as $\ell_1$-based relaxations such as the Lasso have desirable properties for sparse parametric models, similar $\ell_1$-based approaches have proven to be successful. Ravikumar et al. [12] propose a back-fitting algorithm to recover the component functions $h_j$ and prove consistency in both subset recovery and consistency in empirical $L^2(\mathbb{P}_n)$ norm. Meier et al. [9] propose a method that involves a sparsity-smoothness penalty term, and also demonstrate consistency in $L^2(\mathbb{P})$ norm. In the special case that $\mathcal{H}$ is a reproducing kernel Hilbert space (RKHS), Koltchinskii and Yuan [7] analyze a least-squares estimator based on imposing an $\ell_1 - \ell_{\mathcal{H}}$-penalty. The analysis in these paper demonstrates that under certain conditions on the covariates, such regularized procedures can yield estimators that are consistent in the $L^2(\mathbb{P})$-norm even when $n \ll p$.

Of complementary interest to the rates achievable by practical methods are the fundamental limits of the estimating sparse additive models, meaning lower bounds that apply to any algorithm. Although such lower bounds are well-known under classical scaling (where $p$ remains fixed independent of $n$), to the best of our knowledge, lower bounds for minimax rates on sparse additive models have not been determined. In this paper, our main result is to establish a lower bound on the minimax rate in $L^2(\mathbb{P})$ norm that scales as $\max\left(\frac{s \log(p/s)}{n}, s\epsilon_n^2(\mathcal{H})\right)$. The first term $\frac{s \log(p/s)}{n}$ is a *subset selection term*, independent of the univariate function space $\mathcal{H}$ in which the additive components lie, that reflects the difficulty of finding the subset $S$. The second term $s\epsilon_n^2(\mathcal{H})$ in an *s-dimensional estimation term*, which depends on the low dimension $s$ but not the ambient dimension $p$, and reflects the difficulty of estimating the sum of $s$ univariate functions, each drawn from function class $\mathcal{H}$. Either the subset selection or $s$-dimensional estimation term dominates, depending on the relative sizes of $n$, $p$, and $s$ as well as $\mathcal{H}$. Importantly, our analysis applies both in the low-dimensional setting ($n \gg p$) and the high-dimensional setting ($p \gg n$) provided that $n, p$ and $s$ are going to $\infty$. Our analysis is based on information-theoretic techniques centered around the use of metric entropy, mutual information and Fano's inequality in order to obtain lower bounds. Such techniques are standard in the analysis of non-parametric procedures under classical scaling [5, 2, 17], and have also been used more recently to develop lower bounds for high-dimensional inference problems [16, 11].

The remainder of the paper is organized as follows. In the next section, the results are stated including appropriate preliminary concepts, notation and assumptions. In Section 3, we state the main results, and provide some comparisons to the rates achieved by existing algorithms. In Section 4, we provide an overview of the proof. We discuss and summarize the main consequences in Section 5.

## 2 Background and problem formulation

In this paper, we consider a non-parametric regression model with random design, meaning that we make $n$ observations of the form

$$y^{(i)} = f^*(X^{(i)}) + w^{(i)}, \qquad \text{for } i = 1, 2, \ldots, n. \tag{2}$$

Here the random vectors $X^{(i)} \in \mathbb{R}^p$ are the covariates, and have elements $X_j^{(i)}$ drawn i.i.d. from some underlying distribution $\mathbb{P}$. We assume that the noise variables $w^{(i)} \sim \mathcal{N}(0, \sigma^2)$ are drawn independently, and independent of all $X^{(i)}$'s. Given a base class $\mathcal{H}$ of univariate functions with norm $\|\cdot\|_{\mathcal{H}}$, consider the class of functions $f : \mathbb{R}^p \to \mathbb{R}$ that have an additive decomposition:

$$\mathcal{F} := \big\{ f : \mathbb{R}^p \to \mathbb{R} \mid f(x_1, x_2, ..., x_p) = \sum_{j=1}^{p} h_j(x_j), \quad \text{and} \quad \|h_j\|_{\mathcal{H}} \leq 1 \quad \forall j = 1, \ldots, p \big\}.$$

Given some integer $s \in \{1, \ldots, p\}$, we define the function class $\mathcal{F}_0(s)$, which is a union of $\binom{p}{s}$ $s$-dimensional subspaces of $\mathcal{F}$, given by

$$\mathcal{F}_0(s) := \big\{ f \in \mathcal{F} \mid \sum_{j=1}^{p} \mathbb{I}(h_j \neq 0) \leq s \big\}. \tag{3}$$

The *minimax rate* of estimation over $\mathcal{F}_0(s)$ is defined by the quantity $\min_{\widehat{f}} \max_{f^* \in \mathcal{F}_0(s)} \mathbb{E}\|\widehat{f} - f^*\|_{L^2(\mathbb{P})}^2$, where the expectation is taken over the noise $w$, and randomness in the sampling, and $\widehat{f}$ ranges over all (measurable)

functions of the observations $\{(y^{(i)}, X^{(i)})\}_{i=1}^n$. The goal of this paper is to determine lower bounds on this minimax rate.

## 2.1 Inner products and norms

Given a univariate function $h_j \in \mathcal{H}$, we define the usual $L^2(\mathbb{P})$ inner product

$$\langle h_j, h_j' \rangle_{L^2(\mathbb{P})} := \int_{\mathbb{R}} h_j(x) h_j'(x) \, d\mathbb{P}(x).$$

(With a slight abuse of notation, we use $\mathbb{P}$ to refer to the measure over $\mathbb{R}^p$ as well as the induced marginal measure in each direction defined over $\mathbb{R}$). Without loss of generality (re-centering the functions as needed), we may assume

$$\mathbb{E}[h_j(X)] = \int_{\mathbb{R}} h_j(x) \, d\mathbb{P}(x) = 0,$$

for all $h_j \in \mathcal{H}$. As a consequence, we have $\mathbb{E}[f(X_1, \ldots, X_p)] = 0$ for all functions $f \in \mathcal{F}_0(s)$. Given our assumption that the covariate vector $X = (X_1, \ldots, X_p)$ has independent components, the $L^2(\mathbb{P})$ inner product on $\mathcal{F}$ has the additive decomposition $\langle f, f' \rangle_{L^2(\mathbb{P})} = \sum_{j=1}^p \langle h_j, h_j' \rangle_{L^2(\mathbb{P})}$. (Note that if independence were not assumed the $L^2(\mathbb{P})$ inner product over $\mathcal{F}$ would involve cross-terms.)

## 2.2 Kullback-Leibler divergence

Since we are using information theoretic techniques, we will be using the Kullback-Leibler (KL) divergence as a measure of "distance" between distributions. For a given pair of functions $f$ and $\widetilde{f}$, consider the $n$-dimensional vectors $f(X) = \big(f(X^{(1)}), f(X^{(2)}), \ldots, f(X^{(n)})\big)^T$ and $\widetilde{f}(X) = \big(\widetilde{f}(X^{(1)}), \widetilde{f}(X^{(2)}), \ldots, \widetilde{f}(X^{(n)})\big)^T$. Since $Y|f(X) \sim \mathcal{N}(f(X), \sigma^2 I_{n \times n})$ and $Y|\widetilde{f}(X) \sim \mathcal{N}(\widetilde{f}(X), \sigma^2 I_{n \times n})$,

$$D(Y|f(X) \, \| \, Y|\widetilde{f}(X)) = \frac{1}{2\sigma^2} \| f(X) - \widetilde{f}(X) \|_2^2. \tag{4}$$

We also use the notation $D(f \, \| \, \widetilde{f})$ to mean the average K-L divergence between the distributions of $Y$ induced by the functions $f$ and $\widetilde{f}$ respectively. Therefore we have the relation

$$
\begin{aligned}
D(f \, \| \, \widetilde{f}) &= \mathbb{E}_X\big[D(Y|f(X) \, \| \, Y|\widetilde{f}(X))\big] \\
&= \frac{n}{2\sigma^2} \| f - \widetilde{f} \|_{L^2(\mathbb{P})}^2.
\end{aligned}
\tag{5}
$$

This relation between average K-L divergence and squared $L^2(\mathbb{P})$ distance plays an important role in our proof.

## 2.3 Metric entropy for function classes

In this section, we define the notion of metric entropy, which provides a way in which to measure the relative sizes of different function classes with respect to some metric $\rho$. More specifically, central to our results is the metric entropy of $\mathcal{F}_0(s)$ with respect to the $L^2(\mathbb{P})$ norm.

**Definition 1** (Covering and packing numbers). Consider a metric space consisting of a set $\mathcal{S}$ and a metric $\rho : \mathcal{S} \times \mathcal{S} \to \mathbb{R}_+$.

(a) An $\epsilon$-covering of $\mathcal{S}$ in the metric $\rho$ is a collection $\{f^1, \ldots, f^N\} \subset \mathcal{S}$ such that for all $f \in \mathcal{S}$, there exists some $i \in \{1, \ldots, N\}$ with $\rho(f, f^i) \leq \epsilon$. The $\epsilon$-covering number $N_\rho(\epsilon)$ is the cardinality of the smallest $\epsilon$-covering.

(b) An $\epsilon$-packing of $\mathcal{S}$ in the metric $\rho$ is a collection $\{f^1, \ldots, f^M\} \subset \mathcal{S}$ such that $\rho(f^i, f^j) \geq \epsilon$ for all $i \neq j$. The $\epsilon$-packing number $M_\rho(\epsilon)$ is the cardinality of the largest $\epsilon$-packing.

The covering and packing entropy (denoted by $\log N_\rho(\epsilon)$ and $\log M_\rho(\epsilon)$ respectively) are simply the logarithms of the covering and packing numbers, respectively. It can be shown that for any convex set, the quantities $\log N_\rho(\epsilon)$ and $\log M_\rho(\epsilon)$ are of the same order (within constant factors independent of $\epsilon$).

In this paper, we are interested in packing (and covering) subsets of the function class $\mathcal{F}_0(s)$ in the $L^2(\mathbb{P})$ metric, and so drop the subscript $\rho$ from here onwards. En route to characterizing the metric entropy of $\mathcal{F}_0(s)$, we need to understand the metric entropy of the unit balls of our univariate function class $\mathcal{H}$—namely, the sets

$$\mathbb{B}_{\mathcal{H}}(1) := \{h \in \mathcal{H} \mid \|h\|_{\mathcal{H}} \le 1\}.$$

The metric entropy (both covering and packing entropy) for many classes of functions are known. We provide some concrete examples here:

(i) Consider the class $\mathcal{H} = \{h_\beta : \mathbb{R} \to \mathbb{R} \mid h_\beta(x) = \beta x\}$ of all univariate linear functions with the norm $\|h_\beta\|_{\mathcal{H}} = |\beta|$. Then it is known [15] that the metric entropy of $\mathbb{B}_{\mathcal{H}}(1)$ scales as $\log M(\epsilon; \mathcal{H}) \sim \log(1/\epsilon)$.

(ii) Consider the class $\mathcal{H} = \{h : [0,1] \to [0,1] \mid |h(x) - h(y)| \le |x - y|\}$ of all 1-Lipschitz functions on $[0,1]$ with the norm $\|h\|_{\mathcal{H}} = \sup_{x \in [0,1]} |h(x)|$. In this case, it is known [15] that the metric entropy scales as $\log M^{\mathcal{H}}(\epsilon; \mathcal{H}) \sim 1/\epsilon$. Compared to the previous example of linear models, note that the metric entropy grows much faster as $\epsilon \to 0$, indicating that the class of Lipschitz functions is much richer.

(iii) Consider the class of Sobolev spaces $W^m$ for $m \ge 1$, consisting of all functions that have $m$ derivatives, and the $m^{th}$ derivative is bounded in $L^2(\mathbb{P})$ norm. In this case, it is known that $\log M(\epsilon; \mathcal{H}) \sim \epsilon^{-\frac{1}{m}}$ (e.g., [3]). Clearly, increasing the smoothness constraint $m$ leads to smaller classes. Such Sobolev spaces are a particular class of functions whose packing/covering entropy grows at a rate polynomial in $\frac{1}{\epsilon}$.

In our analysis, we require that the metric entropy of $\mathbb{B}_{\mathcal{H}}(1)$ satisfy the following technical condition:

**Assumption 1.** Using $\log M(\epsilon; \mathcal{H})$ to denote the packing entropy of the unit ball $\mathbb{B}_{\mathcal{H}}(1)$ in the $L^2(\mathbb{P})$-norm, assume that there exists some $\alpha \in (0,1)$ such that

$$\lim_{\epsilon \to 0} \frac{\log M(\alpha \epsilon; \mathcal{H})}{\log M(\epsilon; \mathcal{H})} > 1.$$

The condition is required to ensure that $\log M(c\epsilon)/\log M(\epsilon)$ can be made arbitrarily small or large uniformly over small $\epsilon$ by changing $c$, so that a bound due to Yang and Barron [17] can be applied. It is satisfied for most non-parametric classes, including (for instance) the Lipschitz and Sobolev classes defined in Examples (ii) and (iii) above. It may fail to hold for certain parametric classes, such as the set of linear functions considered in Example (i); however, we can use an alternative technique to derive bounds for the parametric case (see Corollary 2).

## 3 Main result and some consequences

In this section, we state our main result and then develop some of its consequences. We begin with a theorem that covers the function class $\mathcal{F}_0(s)$ in which the univariate function classes $\mathcal{H}$ have metric entropy that satisfies Assumption 1. We state a corollary for the special cases of univariate classes $\mathcal{H}$ with metric entropy growing polynomial in $(1/\epsilon)$, and also a corollary for the special case of sparse linear regression.

Consider the observation model (2) where the covariate vectors have i.i.d. elements $X_j \sim \mathbb{P}$, and the regression function $f^* \in \mathcal{F}_0(s)$. Suppose that the univariate function class $\mathcal{H}$ that underlies $\mathcal{F}_0(s)$ satisfies Assumption 1. Under these conditions, we have the following result:

**Theorem 1.** *Given $n$ i.i.d. samples from the sparse additive model* (2)*, the minimax risk in squared-$L^2(\mathbb{P})$ norm is lower bounded as*

$$\min_{\widehat{f}} \max_{f^* \in \mathcal{F}_0(s)} \mathbb{E}\|\widehat{f} - f^*\|_{L^2(\mathbb{P})}^2 \ge \max\left[\frac{\sigma^2 s \log(p/s)}{32n}, \; \frac{s}{16}\epsilon_n^2(\mathcal{H})\right], \tag{6}$$

*where, for a fixed constant $c$, the quantity $\epsilon_n(\mathcal{H}) = \epsilon_n > 0$ is largest positive number satisfying the inequality*

$$\frac{n\epsilon_n^2}{2\sigma^2} \le \log M(c\,\epsilon_n). \tag{7}$$

For the case where $\mathcal{H}$ has an entropy that is growing to $\infty$ at a polynomial rate as $\epsilon \to 0$—say $\log M(\epsilon; \mathcal{H}) = \Theta(\epsilon^{-1/m})$ for some $m > \frac{1}{2}$, we can compute the rate for the $s$-dimensional estimation term explicitly.

**Corollary 1.** *For the sparse additive model* (2) *with univariate function space $\mathcal{H}$ such that such that $\log M(\epsilon; \mathcal{H}) = \Theta(\epsilon^{-1/m})$, we have*

$$\min_{\widehat{f}} \max_{f^* \in \mathcal{F}_0(s)} \mathbb{E}\|\widehat{f} - f^*\|^2_{L^2(\mathbb{P})} \geq \max \left[ \frac{\sigma^2 s \log(p/s)}{32n}, C s \left(\frac{\sigma^2}{n}\right)^{\frac{2m}{2m+1}} \right], \tag{8}$$

*for some $C > 0$.*

### 3.1 Some consequences

In this section, we discuss some consequences of our results.

*Effect of smoothness:* Focusing on Corollary 1, for spaces with $m$ bounded derivatives (i.e., functions in the Sobolev space $W^m$), the minimax rate is $n^{-\frac{2m}{2m+1}}$ (for details, see e.g. Stone [13]). Clearly, faster rates are obtained for larger smoothness indices $m$, and as $m \to \infty$, the rate approaches the parametric rate of $n^{-1}$. Since we are estimating over an $s$-dimensional space (under the assumption of independence), we are effectively estimating $s$ univariate functions, each lying within the function space $\mathcal{H}$. Therefore the uni-dimensional rate is multiplied by $s$.

*Smoothness versus sparsity:* It is worth noting that depending on the relative scalings of $s$, $n$ and $p$ and the metric entropy of $\mathcal{H}$, it is possible for either the subset selection term or $s$-dimensional estimation term to dominate the lower bound. In general, if $\frac{\log(p/s)}{n} = o(\epsilon_n^2(\mathcal{H}))$, the $s$-dimensional estimation term dominates, and vice versa (at the boundary, either term determines the minimax rate). In the case of a univariate function class $\mathcal{H}$ with polynomial entropy as in Corollary 1, it can be seen that for $n = o((\log(p/s))^{2m+1})$, the $s$-dimensional estimation term dominates while for $n = \Omega((\log(p/s))^{2m+1})$, the subset selection term dominates.

*Rates for linear models:* Using an alternative proof technique (not the one used in this paper), it is possible [11] to derive the exact minimax rate for estimation in the *sparse linear regression model*, in which we observe

$$y^{(i)} = \sum_{j \in S} \beta_j X_j^{(i)} + w^{(i)}, \quad \text{for } i = 1, 2, ..., n. \tag{9}$$

Note that this is a special case of the general model (2) in which $\mathcal{H}$ corresponds to the class of univariate linear functions (see Example (i)).

**Corollary 2.** *For sparse linear regression model* (9)*, the the minimax rate scales as* $\max \left( \frac{s \log(p/s)}{n}, \frac{s}{n} \right)$.

In this case, we see clearly the subset selection term dominates for $p \to \infty$, meaning the subset selection problem is always "harder" (in a statistical sense) than the $s$-dimensional estimation problem. As shown by Bickel et al. [1], the rate achieved by $\ell_1$-regularized methods is $\frac{s \log p}{n}$ under suitable conditions on the covariates $X$.

*Upper bounds:* To show that the lower bounds are tight, upper bounds that are matching need to be derived. Upper bounds (matching up to constant factors) can be derived via a classical information-theoretic approach (e.g., [5, 2]), which involves constructing an estimator based on a covering set and bounding the covering entropy of $\mathcal{F}_0(s)$. While this estimation approach does not lead to an implementable algorithm, it is a simple theoretical device to demonstrate that lower bounds are tight. We turn our focus on implementable algorithms in the next point.

*Comparison to existing bounds:* We now provide a brief comparison of the minimax lower bounds with upper bounds on rates achieved by existing implementable algorithms provided by past work [12, 7, 9]. Ravikumar et al. [12] propose a back-fitting algorithm to minimize the least-squares objective with a sparsity constraint on the the function $f$. The rates derived in Koltchinskii and Yuan [7] do not match the lower bounds derived in Theorem 1. Further, it is difficult to directly compare the rates in Ravikumar et al. [12] and Meier et al. [9] with our minimax lower bounds since their analysis does not explicitly track the sparsity index $s$. We are currently in the process of conducting a thorough comparison with the above-mentioned $\ell_1$-based methods.

## 4 Proof outline

In this section, we provide an outline of the proof of Theorem 1; due to space constraints, we defer some of the technical details to the full-length version. The proof is based on a combination of information-theoretic

techniques and the concepts of packing and covering entropy, as defined previously in Section 2.3. First, we provide a high-level overview of the proof. The basic idea is to carefully choose two subsets $T_1$ and $T_2$ of the function class $\mathcal{F}_0(s)$ and lower bound the minimax rates over these two subsets. In Section 4.1, application of the generalized Fano method—a technique based on Fano's inequality—to the set $T_1$ defined in equation (10) yields a lower bound on the subset selection term. In Section 4.2, we apply an alternative method for obtaining lower bounds over a second set $T_2$ defined in equation (11) that captures the difficulty of estimating the sum of $s$ univariate functions.. The second technique also exploits Fano's inequality but uses a more refined upper bound on the mutual information developed by Yang and Barron [17].

Before proceding, we first note that for any $T \subset \mathcal{F}_0(s)$, we have

$$\min_{\widehat{f}} \max_{f^* \in \mathcal{F}_0(s)} \mathbb{E}\|\widehat{f} - f^*\|^2_{L^2(\mathbb{P})} \geq \min_{\widehat{f}} \max_{f^* \in T} \mathbb{E}\|\widehat{f} - f^*\|^2_{L^2(\mathbb{P})}.$$

Moreover, for any subsets $T_1, T_2 \subset \mathcal{F}_0(s)$, we have

$$\min_{\widehat{f}} \max_{f^* \in \mathcal{F}_0(s)} \mathbb{E}\|\widehat{f} - f^*\|^2_{L^2(\mathbb{P})} \geq \max\big( \min_{\widehat{f}} \max_{f^* \in T_1} \mathbb{E}\|\widehat{f} - f^*\|^2_{L^2(\mathbb{P})}, \min_{\widehat{f}} \max_{f^* \in T_2} \mathbb{E}\|\widehat{f} - f^*\|^2_{L^2(\mathbb{P})}\big),$$

since the bound holds for each of the two terms. We apply this lower bound using the subsets $T_1$ and $T_2$ defined in equations (10) and (11).

## 4.1 Bounding the complexity of subset selection

For part of the proof, we use the generalized Fano's method [4], which we state below without proof. Given some parameter space, we let $d$ be a metric on it.

**Lemma 1. (Generalized Fano Method)** *For a given integer $r \geq 2$, consider a collection $\mathcal{M}_r = \{\mathbb{P}_1, \ldots, \mathbb{P}_r\}$ of $r$ probability distributions such that*

$$d(\theta(\mathbb{P}_i), \theta(\mathbb{P}_j)) \geq \alpha_r \quad \text{for all } i \neq j,$$

*and the pairwise KL divergence satisfies*

$$D(\mathbb{P}_i \,\|\, \mathbb{P}_j) \leq \beta_r \quad \text{for all } i, j = 1, \ldots, r.$$

*Then the minimax risk over the family is lower bounded as*

$$\max_j \mathbb{E}_j d(\theta(\mathbb{P}_j), \widehat{\theta}) \geq \frac{\alpha_r}{2}\left(1 - \frac{\beta_r + \log 2}{\log r}\right).$$

The proof of Lemma 1 involves applying Fano's inequality over the discrete set of parameters $\theta \in \Theta$ indexed by the set of distributions $\mathcal{M}_r$. Now we construct the set $T_1$ which creates the set of probability distributions $\mathcal{M}_r$.

Let $g$ be an arbitrary function in $\mathcal{H}$ such that $\|g\|_{L^2(\mathbb{P})} = \frac{\sigma}{4}\sqrt{\frac{\log (p/s)}{n}}$. The set $T_1$ is defined as

$$T_1 := \left\{ f : f(X_1, X_2, ..., X_p) = \sum_{j=1}^{p} c_j g(X_j), c_j \in \{-1, 0, 1\} \mid \|c\|_0 = s \right\}. \tag{10}$$

$T_1$ may be viewed as a hypercube of $\mathcal{F}_0(s)$ and will lead to the lower bound for the 'subset selection' term. This hypercube construction is often used to prove lower bounds (see Yu [18]). Next, we require a further reduction of the set $T_1$ to a set $A$ (defined in Lemma 2) to ensure that elements of $A$ are well-separated in $L^2(\mathbb{P})$ norm. The construction of $A$ is as follows:

**Lemma 2.** *There exists a subset $A \subset T_1$ such that:*
*(i) $\log |A| \geq \frac{1}{2}s \log(p/s)$,*
*(ii) $\|f - f'\|^2_{L^2(\mathbb{P})} \geq \frac{\sigma^2 s \log(p/s)}{16n} \; \forall \, f, \; f' \in A$, and*
*(iii) $D(f \,\|\, f') \leq \frac{1}{8}s \log(p/s) \; \forall f, \; f' \in A$.*

The proof involves using a combinatoric argument to construct the set $A$. For an argument on how the set is constructed, see Kühn [8]. For $s \log \frac{p}{s} \geq 8 \log 2$, applying the Generalized Fano Method (Lemma 1) together with Lemma 2 yields the bound

$$\min_{\widehat{f}} \max_{f^* \in \mathcal{F}_0(s)} \mathbb{E}\|\widehat{f} - f^*\|^2_{L^2(\mathbb{P})} \geq \min_{\widehat{f}} \max_{f^* \in A} \mathbb{E}\|\widehat{f} - f^*\|^2_{L^2(\mathbb{P})} \geq \frac{\sigma^2 s \log(p/s)}{32n}.$$

This completes the proof for the subset selection term ($\frac{s \log (p/s)}{n}$) in Theorem 1.

## 4.2 Bounding the complexity of $s$-dimensional estimation

Next we derive a bound for the $s$-dimensional estimation term by determining a lower bound over $T_2$. Let $S$ be an arbitrary subset of $s$ integers in $\{1, 2, .., p\}$, and define the set $\mathcal{F}_S$ as

$$T_2 := \mathcal{F}_S := \big\{ f \in \mathcal{F} \; : \; f(X) = \sum_{j \in S} h_j(X_j) \big\}. \tag{11}$$

Clearly $\mathcal{F}_S \subset \mathcal{F}_0(s)$ meaning that

$$\min_{\widehat{f}} \max_{f^* \in \mathcal{F}_0(s)} \mathbb{E}\|\widehat{f} - f^*\|_{L^2(\mathbb{P})}^2 \geq \min_{\widehat{f}} \max_{f^* \in \mathcal{F}_S} \mathbb{E}\|\widehat{f} - f^*\|_{L^2(\mathbb{P})}^2.$$

We use a technique used in Yang and Barron [17] to lower bound the minimax rate over $\mathcal{F}_S$. The idea is to construct a maximal $\delta_n$-packing set for $\mathcal{F}_S$ and a minimal $\epsilon_n$-covering set for $\mathcal{F}_S$, and then to apply Fano's inequality to a carefully chosen mixture distribution involving the covering and packing sets (see the full-length version for details). Following these steps yields the following result:

**Lemma 3.**
$$\min_{\widehat{f}} \max_{f^* \in \mathcal{F}_S} \mathbb{E}\|\widehat{f} - f^*\|_{L^2(\mathbb{P})}^2 \geq \frac{\delta_n^2}{4}\left(1 - \frac{\log N(\epsilon_n; \mathcal{F}_S) + n\epsilon_n^2/2\sigma^2 + \log 2}{\log M(\delta_n; \mathcal{F}_S)}\right).$$

Now we have a bound with expressions involving the covering and packing entropies of the $s$-dimensional space $\mathcal{F}_S$. The following Lemma allows bounds on $\log M(\epsilon; \mathcal{F}_S)$ and $\log N(\epsilon; \mathcal{F}_S)$ in terms of the unidimensional packing and covering entropies respectively:

**Lemma 4.** *Let $\mathcal{H}$ be function space with a packing entropy $\log M(\epsilon; \mathcal{H})$ that satisfies Assumption 1. Then we have the bounds*

$$\log M(\epsilon; \mathcal{F}_S) \geq s \log M(\epsilon/\sqrt{s}; \mathcal{H}), \quad and \quad \log N(\epsilon; \mathcal{F}_S) \leq s \log N(\epsilon/\sqrt{s}; \mathcal{H}).$$

The proof involves constructing $\frac{\epsilon}{\sqrt{s}}$- packing set and covering sets in each of the $s$ dimensions and displaying that these are $\epsilon$-packing and coverings sets in $\mathcal{F}_S$ (respectively). Combining Lemmas 3 and 4 leads to the inequality

$$\min_{\widehat{f}} \max_{f^* \in \mathcal{F}_S} \mathbb{E}\|\widehat{f} - f^*\|_{L^2(\mathbb{P})}^2 \geq \frac{\delta_n^2}{4}\left(1 - \frac{s \log N(\epsilon_n/\sqrt{s}; \mathcal{H}) + n\epsilon_n^2/2\sigma^2 + \log 2}{s \log M(\delta_n/\sqrt{s}; \mathcal{H})}\right). \tag{12}$$

Now we choose $\epsilon_n$ and $\delta_n$ to meet the following constraints:

$$\frac{n}{2\sigma^2}\epsilon_n^2 \;\leq\; s \log N(\frac{\epsilon_n}{\sqrt{s}}; \mathcal{H}), \quad \text{and} \tag{13a}$$

$$4 \log N(\frac{\epsilon_n}{\sqrt{s}}; \mathcal{H}) \;\leq\; \log M(\frac{\delta_n}{\sqrt{s}}; \mathcal{H}). \tag{13b}$$

Combining Assumption 1 with the well-known relations $\log M(2\epsilon; \mathcal{H}) \leq \log N(2\epsilon; \mathcal{H}) \leq \log M(\epsilon; \mathcal{H})$, we conclude that in order to satisfy inequalities (13a) and (13b), it is sufficient to choose $\epsilon_n = c\delta_n$ for a constant $c$, and then require that $s \log M(\frac{c\delta_n}{\sqrt{s}}; \mathcal{H}) \geq \frac{n\delta_n^2}{2\sigma^2}$. Furthermore, if we define $\delta_n/\sqrt{s} = \widetilde{\delta}_n$, then this inequality can be re-expressed as $\log M(c\widetilde{\delta}_n) \geq \frac{n\widetilde{\delta}_n^2}{2\sigma^2}$. For $\frac{n}{2\sigma^2}\epsilon_n^2 \geq \log 2$, using inequalities (13a) and (13b) together with equation (12) yields the desired rate

$$\min_{\widehat{f}} \max_{f^* \in \mathcal{F}_S} \mathbb{E}\|\widehat{f} - f^*\|_{L^2(\mathbb{P})}^2 \geq \frac{s\widetilde{\delta}_n^2}{16},$$

thereby completing the proof.

## 5    Discussion

In this paper, we have derived lower bounds for the minimax risk in squared $L^2(\mathbb{P})$ error for estimating sparse additive models based on the sum of univariate functions from a function class $\mathcal{H}$. The rates show that the estimation problem effectively decomposes into a *subset selection problem* and an *s-dimensional estimation*

*problem*, and the "harder" of the two problems (in a statistical sense) determines the rate of convergence. More concretely, we demonstrated that the subset selection term scales as $\frac{s \log(p/s)}{n}$, depending linearly on the number of components $s$ and only logarithmically in the ambient dimension $p$. This subset selection term is independent of the univariate function space $\mathcal{H}$. On the other hand, the $s$-dimensional estimation term depends on the "richness" of the univariate function class, measured by its metric entropy; it scales linearly with $s$ and is independent of $p$. Ongoing work suggests that our lower bounds are tight in many cases, meaning that the rates derived in Theorem 1 are minimax optimal for many function classes.

There are a number of ways in which the work can be extended. One implicit and strong assumption in our analysis was that the covariates $X_j, j = 1, 2, ..., p$ are independent. It would be interesting to investigate the case when the random variables are endowed with some correlation structure. One would expect the rates to change, particularly if many of the variables are collinear. It would also be interesting to develop a more complete understanding of whether computationally efficient algorithms [7, 12, 9] based on regularization achieve the lower bounds on the minimax rate derived in this paper.

## References

[1] P. Bickel, Y. Ritov, and A. Tsybakov. Simultaneous analysis of the Lasso and Dantzig selector. *Annals of Statistics*, 2009. To appear.

[2] L. Birgé. Approximation dans les espaces metriques et theorie de l'estimation. *Z. Wahrsch. verw. Gebiete*, 65:181–327, 1983.

[3] M. S. Birman and M. Z. Solomjak. Piecewise-polynomial approximations of functions of the classes $W_p^\alpha$. *Math. USSR-Sbornik*, 2(3):295–317, 1967.

[4] T. S. Han and S. Verdu. Generalizing the Fano inequality. *IEEE Transactions on Information Theory*, 40:1247–1251, 1994.

[5] R. Z. Has'minskii. A lower bound on the risks of nonparametric estimates of densities in the uniform metric. *Theory Prob. Appl.*, 23:794–798, 1978.

[6] T. Hastie and R. Tibshirani. *Generalized Additive Models*. Chapman and Hall Ltd, Boca Raton, 1999.

[7] V. Koltchinskii and M. Yuan. Sparse recovery in large ensembles of kernel machines. In *Proceedings of COLT*, 2008.

[8] T. Kühn. A lower estimate for entropy numbers. *Journal of Approximation Theory*, 110:120–124, 2001.

[9] L. Meier, S. van de Geer, and P. Buhlmann. High-dimensional additive modeling. *Annals of Statistics*, To appear.

[10] N. Meinshausen and B.Yu. Lasso-type recovery of sparse representations for high-dimensional data. *Annals of Statistics*, 37(1):246–270, 2009.

[11] G. Raskutti, M. J. Wainwright, and B. Yu. Minimax rates of estimation for high-dimensional linear regression over $\ell_q$-balls. Technical Report arXiv:0910.2042, UC Berkeley, Department of Statistics, 2009.

[12] P. Ravikumar, H. Liu, J. Lafferty, and L. Wasserman. Sparse additive models. *Journal of the Royal Statistical Society*, To appear.

[13] C. J. Stone. Optimal global rates of convergence for nonparametric regression. *Annals of Statistics*, 10:1040–1053, 1982.

[14] R. Tibshirani. Regression shrinkage and selection via the lasso. *Journal of the Royal Statistical Society, Series B*, 58(1):267–288, 1996.

[15] S. van de Geer. *Empirical Processes in M-Estimation*. Cambridge University Press, 2000.

[16] M. J. Wainwright. Information-theoretic bounds for sparsity recovery in the high-dimensional and noisy setting. *IEEE Trans. Info. Theory*, December 2009. Presented at International Symposium on Information Theory, June 2007.

[17] Y. Yang and A. Barron. Information-theoretic determination of minimax rates of convergence. *Annals of Statistics*, 27(5):1564–1599, 1999.

[18] B. Yu. Assouad, Fano and Le Cam. *Research Papers in Probability and Statistics: Festschrift in Honor of Lucien Le Cam*, pages 423–435, 1996.

[19] C. H. Zhang and J. Huang. The sparsity and bias of the lasso selection in high-dimensional linear regression. *Annals of Statistics*, 36:1567–1594, 2006.

